# Learning Curves, Model Selection and Complexity of Neural Networks

**Noboru Murata**
Department of Mathematical Engineering and Information Physics
University of Tokyo, Tokyo 113, JAPAN
E-mail: `mura@sat.t.u-tokyo.ac.jp`

**Shuji Yoshizawa**
Dept. Mech. Info.
University of Tokyo

**Shun-ichi Amari**
Dept. Math. Eng. and Info. Phys.
University of Tokyo

## Abstract

Learning curves show how a neural network is improved as the number of training examples increases and how it is related to the network complexity. The present paper clarifies asymptotic properties and their relation of two learning curves, one concerning the predictive loss or generalization loss and the other the training loss. The result gives a natural definition of the complexity of a neural network. Moreover, it provides a new criterion of model selection.

## 1   INTRODUCTION

The learning curve shows how well the behavior of a neural network is improved as the number of training examples increases and how it is related with the complexity of neural networks. This provides us with a criterion for choosing an adequate network in relation to the number

of training examples. Some researchers have attacked this problem by using statistical mechanical methods (see Levin et al. [1990], Seung et al. [1991], etc.) and some by information theory and algorithmic methods (see Baum and Haussler

[1989], etc.). The present paper elucidates asymptotic properties of the learning curve from the statistical point of view, giving a new criterion for model selection.

## 2 STATEMENT OF THE PROBLEM

Let us consider a stochastic neural network, which is parameterized by a set of $m$ weights $\theta = (\theta^1, \cdots, \theta^m)$ and whose input-output relation is specified by a conditional probability $p(y|x, \theta)$. In other words, for an input signal is $x \in \mathbf{R}^{n_{in}}$, the probability distribution of output $y \in \mathbf{R}^{n_{out}}$ is given by $p(y|x, \theta)$.

A typical form of the stochastic neural network is as follows: let us consider a multi-layered network $f(x, \theta)$ where $\theta$ is a set of $m$ parameters $\theta = (\theta^1, \cdots, \theta^m)$ and its components correspond to weights and thresholds of the network. When some input $x$ is given, the network produce an output

$$y = f(x, \theta) + \eta(x), \tag{1}$$

where $\eta(x)$ is noise whose conditional distribution is given by $a(\eta|x)$. Then the conditional distribution of the network, which specifies the input-output relation, is given by

$$p(y|x, \theta) = a(y - f(x, \theta)|x). \tag{2}$$

We define a training sample $\xi^t = \{(x_1, y_1), \cdots, (x_t, y_t)\}$ as a set of $t$ examples generated from the true conditional distribution $q(y|x)$, where $x_i$ is generated from a probability distribution $r(x)$ independently. We should note that both $r(x)$ and $q(y|x)$ are unknown and we need not assume the faithfulness of the model, that is, we do not assume that there exists a parameter $\theta^*$ which realize the true distribution $q(y|x)$ such that $p(y|x, \theta^*) = q(y|x)$.

Our purpose is to find an appropriate parameter $\theta$ which realizes a good approximation $p(y|x, \theta)$ to $q(y|x)$. For this purpose, we use a loss function

$$L(\theta) = D(r; q|p(\theta)) + S(\theta) \tag{3}$$

as a criterion to be minimized, where $D(r; q|p(\theta))$ represents a general divergence measure between two conditional probabilities $q(y|x)$ and $p(y|x, \theta)$ in the expectation form under the true input-output probability

$$D(r; q|p(\theta)) = \int r(x)q(y|x)k(x, y, \theta)dxdy \tag{4}$$

and $S(\theta)$ is a regularization term to fit the smoothness condition of outputs (Moody [1992]). So the loss function is rewritten as a expectation form

$$L(\theta) = \int r(x)q(y|x)d(x, y, \theta)dxdy, \quad d(x, y, \theta) = k(x, y, \theta) + S(\theta), \tag{5}$$

and $d(x, y, \theta)$ is called the pointwise loss function.

A typical case of the divergence $D$ of the multi-layered network $f(x, \theta)$ with noise is the squared error

$$D(r; q|p(\theta)) = \int r(x)q(y|x)\|y - f(x, \theta)\|^2 dxdy, \tag{6}$$

The error function of an ordinary multi-layered network is in this form, and the conventional Back-Propagation method is derived from this type of loss function.

Another typical case is the Kullback-Leibler divergence

$$D(r; q|p(\theta)) = \int r(x)q(y|x) \log \frac{q(y|x)}{p(y|x,\theta)} dx dy. \tag{7}$$

The integration $\int r(x)q(y|x) \log q(y|x) dx dy$ is a constant called a conditional entropy, and we usually use the following abbreviated form instead of the previous divergence:

$$D(r; q|p(\theta)) = - \int r(x)q(y|x) \log p(y|x,\theta) dx dy. \tag{8}$$

Next, we define an optimum of the parameter in the sense of the loss function that we introduced. We denote by $\theta^*$ the optimal parameter that minimizes the loss function $L(\theta)$, that is,

$$L(\theta^*) = \min_{\theta} L(\theta), \tag{9}$$

and we regard $p(y|x, \theta^*)$ as the best realization of the model.

When a training sample $\xi^t$ is given, we can also define an empirical loss function:

$$L(\theta) = D(r; \hat{q}|p(\theta)) + S(\theta), \tag{10}$$

where $\hat{r}$, $\hat{q}$ are the empirical distributions given by the sample $\xi^t$, that is,

$$D(r; \hat{q}|p(\theta)) = \frac{1}{t} \sum_{i=1}^{t} k(x_i, y_i, \theta), \quad (x_i, y_i) \in \xi^t. \tag{11}$$

In practical case, we consider the empirical loss function and search for the quasi-optimal parameter $\theta$ defined by

$$L(\theta) = \min_{\theta} L(\theta), \tag{12}$$

because the true distributions $r(x)$ and $q(y|x)$ are unknown and we can only use examples $(x_i, y_i)$ observed from the true distribution $r(x)q(y|x)$. We should note that the quasi-optimal parameter $\theta$ is a random variable depending on the sample $\xi^t$, each element of which is chosen randomly.

The following lemma guarantees that we can use the empirical loss function instead of the actual loss function when the number of examples $t$ is large.

**Lemma 1** *If the number of examples $t$ is large enough, it is shown that the quasi-optimal parameter $\theta$ is normally distributed around the optimal parameter $\theta^*$, that is,*

$$\theta \sim N \left( \theta^*, \frac{1}{t} Q^{-1} G Q^{-1} \right), \tag{13}$$

*where*

$$G = \int r(x)q(y|x) \nabla d(x, y, \theta^*) \nabla d(x, y, \theta^*)^{\mathrm{T}} dx dy, \tag{14}$$

$$Q = \int r(x)q(y|x) \nabla \nabla d(x, y, \theta^*) dx dy, \tag{15}$$

*and $\nabla$ denotes the differential operator with respect to $\theta$.*

This lemma is proved by using the usual statistical methods.

## 3  LEARNING PROCEDURE

In many cases, however, it is difficult to obtain the quasi-optimal parameter $\hat{\theta}$ by minimizing the equation (10) directly. We therefore often use a stochastic descent method to get an approximation to the quasi-optimal parameter $\hat{\theta}$.

**Definition 1 (Stochastic Descent Method)** *In each learning step, an example is re-sampled from the given sample $\xi^t$ randomly, and the following modification is applied to the parameter $\theta_n$ at step $n$,*

$$\theta_{n+1} = \theta_n - \varepsilon \nabla d(x_{i(n)}, y_{i(n)}, \theta_n), \tag{16}$$

*where $\varepsilon$ is a positive value called a learning coefficient and $(x_{i(n)}, y_{i(n)})$ is the re-sampled example at step $n$.*

This is a sequential learning method and the operations of random sampling from $\xi^t$ in each learning step is called *the re-sampling plan*. The parameter $\theta_n$ at step $n$ is a random variable as a function of the re-sampled sequence $\omega = \{(x_{i(1)}, y_{i(1)}), \cdots, (x_{i(n)}, y_{i(n)})\}$. However, if the initial value of $\theta$ is appropriate (this assumption prevents being stuck in local minima) and if the learning step $n$ is large enough, it is shown that the learned parameter $\theta_n$ is normally distributed around the quasi-optimal parameter.

**Lemma 2** *If the learning step $n$ is large enough and the learning coefficient $\varepsilon$ is small enough, the parameter $\theta_n$ is normally distributed asymptotically, that is,*

$$\theta_n \sim N(\theta, \varepsilon V), \tag{17}$$

*where $V$ satisfies the following relation*

$$G = QV + VQ, \tag{18}$$

$$G = \frac{1}{t} \sum_{i=1}^{t} \nabla d(x_i, y_i, \theta) \nabla d(x_i, y_i, \theta)^{\mathrm{T}}, \quad Q = \frac{1}{t} \sum_{i=1}^{t} \nabla \nabla d(x_i, y_i, \theta).$$

In the following discussion, we assume that $n$ is large enough and $\varepsilon$ is small enough, and we denote the learned parameter by

$$\tilde{\theta}(= \theta_n). \tag{19}$$

The distribution of the random variable $\tilde{\theta}$, therefore, can be regarded as the normal distribution $N(\theta, \varepsilon V)$.

## 4  LEARNING CURVES

It is important to evaluate the difference between two quantities $L(\tilde{\theta})$ and $\hat{L}(\tilde{\theta})$. The quantity $L(\tilde{\theta})$ is called the predictive loss or the generalization error, which shows

the average loss of the trained network when a novel example is given. On the other hand, the quantity $\hat{L}(\hat{\theta})$ is called the training loss or the training error, which shows the average loss evaluated by the examples used in training. Since these quantities depend on the sample $\xi^t$ and the re-sampled sequence $\omega$, we take the expectation E and the variance Var with respect to the sample $\xi^t$ and the re-sampling sequence $\omega$.

First, let us consider the predictive loss which is the average loss of the trained network when a new example (which does not belong to the sample $\xi^t$) is given. This averaging operation is replaced by averaging all over the input-output pairs, because the measure of the sample $\xi^t$ is zero. Then the predictive loss is written as

$$L(\tilde{\theta}) = \int r(x)q(y|x)d(x,y,\tilde{\theta})dxdy. \tag{20}$$

From the properties of $\tilde{\theta}$ and $\hat{\theta}$, we can prove the following important relations.

**Theorem 1** *The predictive loss asymptotically satisfies*

$$\mathrm{E}[L(\tilde{\theta})] = L(\theta^*) + \frac{1}{2t}\mathrm{tr}GQ^{-1} + \frac{\varepsilon}{2}\mathrm{tr}QV. \tag{21}$$

$$\mathrm{Var}[L(\tilde{\theta})] = \frac{1}{2t^2}\mathrm{tr}GQ^{-1}GQ^{-1} + \frac{\varepsilon^2}{2}\mathrm{tr}QVQV + \frac{\varepsilon}{t}\mathrm{tr}GV. \tag{22}$$

Roughly speaking, there exist two random values $Y_1$ and $Y_2$, and the predictive loss can be written as the following form:

$$\begin{aligned}L(\tilde{\theta}) &= L(\theta^*) + \frac{1}{2t}\mathrm{tr}GQ^{-1} + \frac{\varepsilon}{2}\mathrm{tr}QV \\ &\quad + \frac{1}{t}Y_1 + \varepsilon Y_2 + o_p(\frac{1}{t}) + o_p(\varepsilon),\end{aligned} \tag{23}$$

where $Y_1$ and $Y_2$ satisfy

$$\mathrm{E}[Y_1] = 0, \qquad \mathrm{Var}[Y_1] = \frac{1}{2}\mathrm{tr}GQ^{-1}GQ^{-1},$$
$$\mathrm{E}[Y_2] = 0, \qquad \mathrm{Var}[Y_2] = \frac{1}{2}\mathrm{tr}QVQV,$$
$$\mathrm{Cov}[Y_1Y_2] = \mathrm{tr}GV,$$

E, Var and Cov denote the expectation, the variance and the covariance respectively.

Next, we consider the training loss, i.e., the average loss evaluated by the examples used in training. Just as we did in the previous theorem, we can get the following relations.

**Theorem 2** *The training loss asymptotically satisfy*

$$\mathrm{E}[L(\tilde{\theta})] = L(\theta^*) - \frac{1}{2t}\mathrm{tr}GQ^{-1} + \frac{\varepsilon}{2}\mathrm{tr}QV, \tag{24}$$

$$\mathrm{Var}[L(\tilde{\theta})] = \frac{1}{t}\left(\int r(x)q(y|x)d(x,y,\theta^*)^2 dxdy \right.$$
$$\left. - \left(\int r(x)q(y|x)d(x,y,\theta^*)dxdy\right)^2\right). \tag{25}$$

Intuitively speaking like the predictive loss, the training loss can be expanded as

$$
\begin{aligned}
\hat{L}(\tilde{\theta}) \;=\; & L(\theta^*) - \frac{1}{2t}\mathrm{tr}GQ^{-1} + \frac{\varepsilon}{2}\mathrm{tr}QV \\
& + \frac{1}{\sqrt{t}}Y_3 + o_p(\frac{1}{\sqrt{t}}) + O_p(\varepsilon),
\end{aligned}
\tag{26}
$$

where $Y_3$ satisfies

$$
\begin{aligned}
\mathrm{E}[Y_3] \;=\;& 0, \\
\mathrm{Var}[Y_3] \;=\;& \int r(x)q(y|x)d(x,y,\theta^*)^2 dxdy - \left(\int r(x)q(y|x)d(x,y,\theta^*)dxdy\right)^2.
\end{aligned}
$$

When we look at two curves $\mathrm{E}[L(\tilde{\theta})]$ and $\mathrm{E}[\hat{L}(\tilde{\theta})]$ as functions of $t$, they are called learning curves which represent the characteristics of learning. The expectations of the predictive loss and the training loss look quite similar. They are different in the sign of the term $1/t$. As the learning coefficient $\varepsilon$ increases, the expectations $\mathrm{E}[L(\tilde{\theta})]$ and $\mathrm{E}[\hat{L}(\tilde{\theta})]$ increase, but as the number of examples $t$ increases, the average predictive loss $\mathrm{E}[L(\tilde{\theta})]$ decreases and the average training loss $\mathrm{E}[\hat{L}(\tilde{\theta})]$ conversely increases. Moreover, their variances are different in the order of $t$. The coefficients $\mathrm{tr}GQ^{-1}$, $\mathrm{tr}QV$, etc. are calculated from the matrices $G$, $Q$ and $V$, which reflect the architecture of the network and the loss criterion to be minimized. We can consider these matrices as representing the complexity of the network. In earlier work, Amari and Murata [1991] introduced an effective complexity of the network, $\mathrm{tr}GQ^{-1}$, by analogy to Akaike's Information Criterion (AIC) (see Akaike [1974]).

## 5 AN APPLICATION FOR MODEL SELECTION

These results naturally leads us to a model selection criterion, which is like the AIC criterion of statistical model selection and which is related those proposed by some researchers (see Murata et al. [1991], Moody [1992]). From the previous relations, we can easily show the following relation

$$
L(\tilde{\theta}) = \hat{L}(\tilde{\theta}) + \frac{1}{t}\mathrm{tr}GQ^{-1} + c,
\tag{27}
$$

where $c$ is a quantity of order $1/\sqrt{t}$ and common to all the networks of the same architecture. We compare the abilities of two different networks, which have the same architecture and are trained by the same sample, but differ in the number of weights or neurons (see Fig.1). We can use a quantity, NIC (Network Information Criterion),

$$
\mathrm{NIC}(\tilde{\theta}) = \hat{L}(\tilde{\theta}) + \frac{1}{t}\mathrm{tr}\tilde{G}\tilde{Q}^{-1},
\tag{28}
$$

where

$$
\tilde{G} = \frac{1}{t}\sum_{i=1}^{t}\nabla d(x_i, y_i, \tilde{\theta})\nabla d(x_i, y_i, \tilde{\theta})^{\mathrm{T}}, \quad \tilde{Q} = \frac{1}{t}\sum_{i=1}^{t}\nabla\nabla d(x_i, y_i, \tilde{\theta}),
\tag{29}
$$

for selecting an optimal network model. Note that this quantity NIC is directly calculable, since all elements of it, $L(\tilde{\theta})$, $\tilde{G}$, $\tilde{Q}$, are given by summing over the

sample $\xi^t$. When we have two models $M_1$ and $M_2$, and the NIC of $M_1$ is smaller than that of $M_2$, the predictive loss of $M_1$ is expected smaller than that of $M_2$, so $M_1$ can be regarded as a better model in the sense of the loss function.

This criterion cannot be used when we compare two networks of different architectures, for example a multi-layered network and a radial basis expansion network. This is because the value $c$ of the order $1/\sqrt{t}$ term is common only to two networks in which one is included in the other as a submodel. The criterion is in general valid only for such a family of networks (see Fig.2).

## 6   CONCLUSIONS

In this paper, we show that there is nice relation between the expectation of the predictive loss and that of the training loss. This result naturally leads us to a new model selection criterion.

We will consider the application of this result as an algorithm for automatically changing the number of hidden units in the learning as future work.

**References**

H. Akaike. (1974) A new look at the statistical model identification. *IEEE Trans. AC*, **19**(6):716–723.

S. Amari. (1967) Theory of adaptive pattern classifiers. *IEEE Trans. EC*, **16**(3):299–307.

S. Amari and N. Murata. (1991) Statistical theory of learning curves under entropic loss criterion. Technical Report METR 91-12, University of Tokyo, Tokyo, Japan.

E. B. Baum and D. Haussler. (1989) What size net gives valid generalization? *Neural Computation*, 1:151–160.

E. Levin, N. Tishby, and S. A. Solla. (1990) A statistical approach to learning and generalization in layered neural networks. *Proc. of IEEE*, **78**(10):1568–1574.

J. E. Moody. (1992) The effective number of parameters: An analysis of generalization and regularization in nonlinear learning systems. In J. E. Moody, S. J. Hanson, and R. P. Lippmann, (eds.), *Advances in Neural Information Processing Systems 4*. San Mateo, CA: Morgan Kaufmann.

N. Murata. (1992) *Statistical asymptotic study on learning* (In Japanese). PhD thesis, University of Tokyo, Tokyo, Japan.

N. Murata, S. Yoshizawa, and S. Amari. (1991) A criterion for determining the number of parameters in an artificial neural network model. In T. Kohonen et al., (eds.), *Artificial Neural Networks*, 9–14. Holland: Elsevier Science Publishers.

H. S. Seung, H. Sompolinsky, and N. Tishby. (1991) Statistical mechanics of learning from examples II. quenched theory and unrealizable rules. Submitted to Physical Review A.

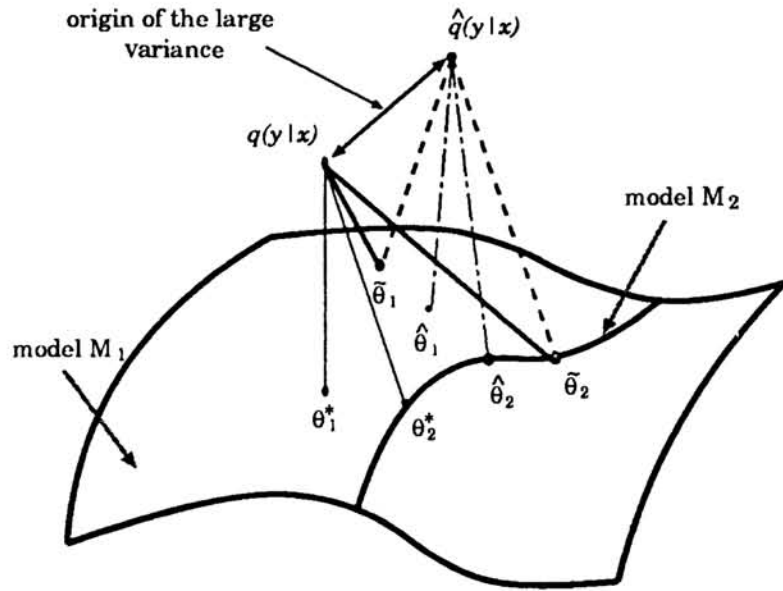

Figure 1: Geometrical representation of hierarchical models: the solid lines between $q(y|x)$ and $\tilde{\theta}_i$ show predictive losses, and the dashed lines between $\hat{q}(y|x)$ and $\hat{\theta}_i$ show training losses. The large variance of the training loss originated in the discrepancy of $q(y|x)$ and $\hat{q}(y|x)$. When we estimate the prediction loss from the training loss, the large variance still remains. But in the case that the model $M_1$ includes the model $M_2$, this variance is common to two models, so we do not have to take care of it.

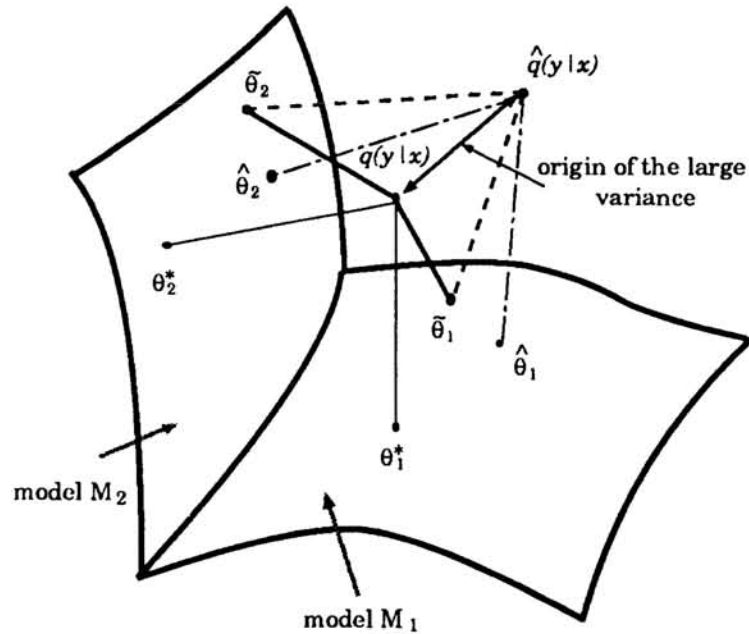

Figure 2: Geometrical representation of non-hierarchical models: the solid lines between $q(y|x)$ and $\tilde{\theta}_i$ show predictive losses, and the dashed lines between $\hat{q}(y|x)$ and $\hat{\theta}_i$ show training losses. The discrepancy of $q(y|x)$ and $\hat{q}(y|x)$ works differently on two models $M_1$ and $M_2$ in estimating predictive losses.